# On the Generalization Ability
# of On-line Learning Algorithms

**Nicolò Cesa-Bianchi**
DTI, University of Milan
via Bramante 65
26013 Crema, Italy
*cesa-bianchi@dti.unimi.it*

**Alex Conconi**
DTI, University of Milan
via Bramante 65
26013 Crema, Italy
*conconi@dti.unimi.it*

**Claudio Gentile**
DSI, University of Milan
via Comelico 39
20135 Milano, Italy
*gentile@dsi.unimi.it*

## Abstract

In this paper we show that on-line algorithms for classification and re-
gression can be naturally used to obtain hypotheses with good data-
dependent tail bounds on their risk. Our results are proven without re-
quiring complicated concentration-of-measure arguments and they hold
for arbitrary on-line learning algorithms. Furthermore, when applied to
concrete on-line algorithms, our results yield tail bounds that in many
cases are comparable or better than the best known bounds.

## 1  Introduction

One of the main contributions of the recent statistical theories for regression and classi-
fication problems [21, 19] is the derivation of functionals of certain empirical quantities
(such as the sample error or the sample margin) that provide uniform risk bounds for all the
hypotheses in a certain class. This approach has some known weak points. First, obtaining
tight uniform risk bounds in terms of meaningful empirical quantities is generally a dif-
ficult task. Second, searching for the hypothesis minimizing a given empirical functional
is often computationally expensive and, furthermore, the minimizing algorithm is seldom
incremental (if new data is added to the training set then the algorithm needs be run again
from scratch).

On-line learning algorithms, such as the Perceptron algorithm [17], the Winnow algo-
rithm [14], and their many variants [16, 6, 13, 10, 2, 9], are general methods for solving
classification and regression problems that can be used in a fully incremental fashion. That
is, they need (in most cases) a short time to process each new training example and adjust
their current hypothesis. While the behavior of these algorithms is well understood in the
so-called *mistake bound* model [14], where no assumptions are made on the way the train-
ing sequence is generated, there are fewer results concerning how to use these algorithms
to obtain hypotheses with small statistical risk.

Littlestone [15] proposed a method for obtaining small risk hypotheses from a run of an
arbitrary on-line algorithm by using a cross validation set to test each one of the hypotheses
generated during the run. This method does not require any convergence property of the on-
line algorithm and provides risk tail bounds that are sharper than those obtainable choosing,
for instance, the hypothesis in the run that survived the longest. Helmbold, Warmuth,

and others [11, 6, 8] showed that, without using any cross-validation sets, one can obtain expected risk bounds (as opposed to the more informative tail bounds) for a hypothesis randomly drawn among those generated during the run.

In this paper we prove, via refinements and extensions of the previous analyses, that on-line algorithms naturally lead to good data-dependent tail bounds without employing the complicated concentration-of-measure machinery needed by other frameworks [19]. In particular we show how to obtain, from an arbitrary on-line algorithm, hypotheses whose risk is close to $m/t$ with high probability (Theorems 2 and 3), where $t$ is the amount of training data and $m$ is a data-dependent quantity measuring the cumulative loss of the on-line algorithm on the actual training data. When applied to concrete algorithms, the loss bound $m$ translates into a function of meaningful data-dependent quantities. For classification problems, the mistake bound for the $p$-norm Perceptron algorithm yields a tail risk bound in terms of the empirical distribution of the margins — see (4). For regression problems, the square loss bound for ridge regression yields a tail risk bound in terms of the eigenvalues of the Gram matrix — see (5).

## 2  Preliminaries and notation

Let $\mathcal{X}, \mathcal{Y}$ be arbitrary sets and $\mathcal{Z} = \mathcal{X} \times \mathcal{Y}$. An *example* is a pair $(x, y)$, where $x$ is an *instance* belonging to $\mathcal{X}$ and $y \in \mathcal{Y}$ is the *label* associated with $x$. Random variables will be denoted in upper case and their realizations will be in lower case. We let $Z$ be the pair of random variables $(X, Y)$, where $X$ and $Y$ take values in $\mathcal{X}$ and $\mathcal{Y}$, respectively. Throughout the paper, we assume that data are generated i.i.d. according to an unknown probability distribution over $\mathcal{Z}$. All probabilities and expectations will be understood with respect to this underlying distribution. We use the short-hand $Z^t$ to denote the vector-valued random variable $(Z_1, \ldots, Z_t)$.

A hypothesis $h$ is any (measurable) mapping from instances $x \in \mathcal{X}$ to predictions $h(x) \in \mathcal{D}$, where $\mathcal{D}$ is a given *decision space*. The *risk* of $h$ is defined by $\mathrm{er}(h) = \mathbb{E}\left[\ell(h(X), Y)\right]$, where $\ell : \mathcal{D} \times \mathcal{Y} \to \mathbb{R}$ is a nonnegative loss function. Unless otherwise specified, we will assume that $\ell$ takes values in $[0, L]$ for some known $0 < L < \infty$. The on-line algorithms we investigate are defined within a well-known mathematical model, which is a generalization of a learning model introduced by Littlestone [14] and Angluin [1]. Let a training sequence $z^t = ((x_1, y_1), \ldots, (x_t, y_t)) \in (\mathcal{X} \times \mathcal{Y})^t$ be fixed. In this learning model, an on-line algorithm processes the examples in $z^t$ one at a time in *trials*, generating a sequence of hypotheses $h_0, h_1, \ldots, h_t$. At the beginning of the $i$-th trial, the algorithm receives the instance $x_i$ and uses its current hypothesis $h_{i-1}$ to compute a prediction $h_{i-1}(x_i) \in \mathcal{D}$ for the label $y_i$ associated with $x_i$. Then, the true value of the label $y_i$ is disclosed and the algorithm suffers a *loss* $\ell(h_{i-1}(x_i), y_i)$, measuring how bad is the prediction $h_{i-1}(x_i)$ for the label $y_i$. Before the next trial begins, the algorithm generates a new hypothesis $h_i$ which may or may not be equal to $h_{i-1}$. We measure the algorithm's performance on $z^t$ by its *cumulative* loss

$$m(z^t) = \sum_{i=1}^{t} \ell(h_{i-1}(x_i), y_i).$$

In our analysis, we will write $M$ and $H_0, \ldots, H_t$ when we want to stress the fact that the cumulative loss and the hypotheses of the on-line algorithm are functions of the random sample $Z^t$. In particular, throughout the paper $H_0$ will denote the (deterministic) initial hypothesis of an arbitrary on-line algorithm and, for each $1 \leq i \leq t$, $H_i$ will be a random variable denoting the $i$-th hypothesis of the on-line algorithm and such that the value of $H_i(Z_1, \ldots, Z_t)$ does not change upon changes in the values of $Z_{i+1}, \ldots, Z_t$.

Our goal is to relate the risk of the hypotheses produced by an on-line algorithm running on an i.i.d. sequence $Z^t$ to the cumulative loss $M(Z^t)$ of the algorithm on that sequence.

The cumulative loss $M(Z^t)$ will be our key empirical (data-dependent) quantity. Via our analysis we will obtain bounds of the form

$$\mathbb{P}\left(\mathrm{er}(f(H_0,\dots,H_t)) \geq \frac{M(Z^t)}{t} + c\sqrt{\frac{1}{t}\ln\frac{1}{\delta}}\right) \leq \delta,$$

where $f(H_0,\dots,H_t)$ is a *specific* function of the sequence of hypotheses $H_0,\dots,H_t$ produced by the algorithm, and $c$ is a suitable positive constant. We will see that for specific on-line algorithms the ratio $M(Z^t)/t$ can be further bounded in terms of meaningful empirical quantities.

Our method centers on the following simple concentration lemma about bounded losses.

**Lemma 1** *Let $\ell$ be an arbitrary bounded loss satisfying $0 \leq \ell \leq L$. Let an arbitrary on-line algorithm output (not necessarily distinct) hypotheses $H_0,\dots,H_t$ when it is run on $Z^t$. Then for any $0 < \delta \leq 1$ we have*

$$\mathbb{P}\left(\frac{1}{t}\sum_{i=1}^{t}\mathrm{er}(H_{i-1}) \geq \frac{M}{t} + L\sqrt{\frac{2}{t}\ln\frac{1}{\delta}}\right) \leq \delta.$$

*Proof.* For each $i = 1,\dots,t$, set $V_{i-1} = \mathrm{er}(H_{i-1}) - \ell(H_{i-1}(X_i), Y_i)$. We have

$$\frac{1}{t}\sum_{i=1}^{t}V_{i-1} = \frac{1}{t}\sum_{i=1}^{t}\mathrm{er}(H_{i-1}) - \frac{M}{t}.$$

Furthermore, $-L \leq V_{i-1} \leq L$, since $\ell$ takes values in $[0, L]$. Also,

$$\mathbb{E}[V_{i-1} \mid \mathcal{F}_{i-1}] = \mathrm{er}(h_{i-1}) - \mathbb{E}[\ell(h_{i-1}(X_i), Y_i) \mid \mathcal{F}_{i-1}], = 0$$

where $\mathcal{F}_{i-1}$ denotes the $\sigma$-algebra generated by $Z_1,\dots,Z_{i-1}$. A direct application of the Hoeffding-Azuma inequality [3] to the bounded random variables $V_0,\dots,V_{t-1}$ proves the lemma. □

## 3  Concentration for convex losses

In this section we investigate the risk of the *average hypothesis*

$$\overline{h} \stackrel{\text{def}}{=} \frac{1}{t}\sum_{i=1}^{t}h_{i-1},$$

where $h_0, h_1,\dots,h_t$ are the hypotheses generated by some on-line algorithm run on $t$ training examples.[1] The average hypothesis generates valid predictions whenever the decision space $\mathcal{D}$ is convex.

**Theorem 2** *Let $\mathcal{D}$ be convex and $\ell : \mathcal{D}\times\mathcal{Y} \to [0, L]$ be convex in the first argument. Let an arbitrary on-line algorithm for $\ell$ output (not necessarily distinct) hypotheses $H_0,\dots,H_t$ when the algorithm is run on $Z^t$. Then for any $0 < \delta < 1$ the following holds*

$$\mathbb{P}\left(\mathrm{er}(\overline{H}) \geq \frac{M}{t} + L\sqrt{\frac{2}{t}\ln\frac{1}{\delta}}\right) \leq \delta.$$

*Proof.* Since $\ell$ is convex in the first argument, by Jensen's inequality we have $\ell\left(\overline{h}(x), y\right) \leq \frac{1}{t}\sum_{i=1}^{t}\ell\left(h_{i-1}(x), y\right)$ . Taking expectation with respect to $(X, Y)$ yields $\text{er}(\overline{h}) \leq \frac{1}{t}\sum_{i=1}^{t}\text{er}(h_{i-1})$. Using the last inequality along with Lemma 1 yields the thesis. $\square$

This theorem, which can be viewed as the tail bound version of the expected bound in [11], implies that the risk of the average hypothesis is close to $m(z^t)/t$ for "most" samples $z^t$. On the other hand, note that it is unlikely that $\sum_{i=1}^{t}\text{er}(H_{i-1})/t$ concentrates around $\mathbb{E}[M]/t$, at least without taking strong assumptions on the underlying on-line algorithm.

An application of Theorem 2 will be shown is Section 5. Here we just note that by applying this theorem to the Weighted Majority algorithm [16], we can prove a version of [5, Theorem 4] for the absolute loss without resorting to sophisticated concentration inequalities (details in the full paper).

## 4  Penalized risk estimation for general losses

If the loss function $\ell$ is nonconvex (such as the 0-1 loss) then the risk of the average hypothesis cannot be bounded in the way shown in the previous section. However, the risk of the best hypothesis, among those generated by the on-line algorithm, cannot be higher than the average risk of the same hypotheses. Hence, Lemma 1 immediately tells us that, under no conditions on the loss function other than boundedness, for most samples $z^t$ at least one of the hypotheses generated has risk close to $m(z^t)/t$. In this section we give a technique (Lemma 4) that, using a penalized risk estimate, finds with high probability such a hypothesis. The argument used is a refinement of Littlestone's method [15]. Unlike Littlestone's, our technique does not require a cross validation set. Therefore we are able to obtain bounds on the risk whose main term is $m(z^t)/t$, where $t$ is the size of the whole set of examples available to the learning algorithm (i.e., training set plus validation set in Littlestone's paper). Similar observations are made in [4], though the analysis there does actually refer only to randomized hypotheses with 0-1 loss (namely, to absolute loss).

Let us define the *penalized risk estimate* of hypothesis $h_i$ by
$$\frac{m_i}{t-i} + c_\delta(t-i) ,$$
where $t-i$ is the length of the suffix $z_{i+1}, \ldots, z_t$ of the training sequence that the on-line algorithm had not seen yet when $h_i$ was generated, $m_i$ is the cumulative loss of $h_i$ on that suffix, and
$$c_\delta(x) = \sqrt{\frac{1}{2\,x}\ln\frac{t(t+1)}{\delta}} .$$
Our algorithm chooses the hypothesis $\widehat{h} = h_{i^*}$, where
$$i^* = \operatorname*{argmin}_{0 \leq i \leq t-1}\left(\frac{m_i}{t-i} + c_\delta(t-i)\right) .$$

For the sake of simplicity, we will restrict to losses $\ell$ with range $[0, 1]$. However, it should be clear that losses taking values in arbitrary bounded real interval can be handled using techniques similar to those shown in Section 3. We prove the following result.

**Theorem 3** *Let an arbitrary on-line algorithm output (not necessarily distinct) hypotheses $H_0, \ldots, H_t$ when it is run on $Z^t$. Then, for any $0 < \delta \leq 1$, the hypothesis $\widehat{H}$ chosen using the penalized risk estimate based on $c_{\delta/2}$ satisfies*
$$\mathbb{P}\left(\text{er}(\widehat{H}) > \frac{M}{t} + 5\sqrt{\frac{1}{t}\ln\frac{2(t+1)}{\delta}}\right) \leq \delta .$$

The proof of this theorem is based on the two following technical lemmas.

**Lemma 4** *Let an arbitrary on-line algorithm output (not necessarily distinct) hypotheses $H_0, \ldots, H_t$ when it is run on $Z^t$. Then for any $0 < \delta < 1$ the following holds:*

$$\mathbb{P}\left(\mathrm{er}(\widehat{H}) > \min_{0 \leq i \leq t-1} \left(\mathrm{er}(H_i) + 2\,c_\delta(t-i)\right)\right) \leq \delta\,.$$

*Proof.* Let $I^* = \mathrm{argmin}_{0 \leq i \leq t-1}\left(\mathrm{er}(H_i) + 2\,c_\delta(t-i)\right)$. Let further $H^* = H_{I^*}$, $M^* = M_{I^*}$, and set for brevity

$$\widehat{e}_i = \frac{M_i}{t-i},$$

$$\widehat{e}^* = \frac{M^*}{t-I^*}.$$

For any fixed $\varepsilon > 0$ we have

$$\mathbb{P}\big(\mathrm{er}(\widehat{H}) > \mathrm{er}(H^*) + \varepsilon\big)$$
$$\leq \sum_{i=0}^{t-1} \mathbb{P}\left(\widehat{e}_i + c_\delta(t-i) \leq \widehat{e}^* + c_\delta(t-I^*)\,,\ \mathrm{er}(H_i) > \mathrm{er}(H^*) + \varepsilon\right). \qquad (1)$$

Now, if

$$\widehat{e}_i + c_\delta(t-i) \leq \widehat{e}^* + c_\delta(t-I^*)$$

holds then either

$$\widehat{e}_i \leq \mathrm{er}(H_i) - c_\delta(t-i)$$

or

$$\widehat{e}^* > \mathrm{er}(H^*) + c_\delta(t-I^*)$$

or

$$\mathrm{er}(H_i) - \mathrm{er}(H^*) < 2\,c_\delta(t-I^*)$$

hold. Hence for any fixed $i$ we can write

$$\mathbb{P}\big(\widehat{e}_i + c_\delta(t-i) \leq \widehat{e}^* + c_\delta(t-I^*)\,,\ \mathrm{er}(H_i) > \mathrm{er}(H^*) + \varepsilon\big)$$
$$\leq \mathbb{P}\left(\widehat{e}_i \leq \mathrm{er}(H_i) - c_\delta(t-i)\,,\ \mathrm{er}(H_i) > \mathrm{er}(H^*) + \varepsilon\right)$$
$$+\,\mathbb{P}\left(\widehat{e}^* > \mathrm{er}(H^*) + c_\delta(t-I^*)\,,\ \mathrm{er}(H_i) > \mathrm{er}(H^*) + \varepsilon\right)$$
$$+\,\mathbb{P}\left(\mathrm{er}(H_i) - \mathrm{er}(H^*) < 2\,c_\delta(t-I^*)\,,\ \mathrm{er}(H_i) > \mathrm{er}(H^*) + \varepsilon\right)$$
$$\leq \mathbb{P}\left(\widehat{e}_i \leq \mathrm{er}(H_i) - c_\delta(t-i)\right) + \mathbb{P}\left(\widehat{e}^* > \mathrm{er}(H^*) + c_\delta(t-I^*)\right) \qquad (2)$$
$$+\,\mathbb{P}\left(\mathrm{er}(H_i) - \mathrm{er}(H^*) < 2\,c_\delta(t-I^*)\,,\ \mathrm{er}(H_i) > \mathrm{er}(H^*) + \varepsilon\right)\,. \qquad (3)$$

Probability (3) is zero if $\varepsilon = 2\,c_\delta(t-I^*)$. Hence, plugging (2) into (1) we can write

$$\mathbb{P}\left(\mathrm{er}(\widehat{H}) > \mathrm{er}(H^*) + 2\,c_\delta(t-I^*)\right)$$
$$\leq \sum_{i=0}^{t-1} \mathbb{P}\left(\widehat{e}_i \leq \mathrm{er}(H_i) - c_\delta(t-i)\right) + t\,\mathbb{P}\left(\widehat{e}^* > \mathrm{er}(H^*) + c_\delta(t-I^*)\right)$$
$$\leq \frac{\delta}{t+1} + t\sum_{i=0}^{t-1} \mathbb{P}\left(\widehat{e}_i \geq \mathrm{er}(H_i) + c_\delta(t-i)\right)$$
$$\leq \frac{\delta}{t+1} + \frac{\delta t}{t+1}$$
$$= \delta,$$

where in the last two inequalities we applied Chernoff-Hoeffding bounds. $\square$

**Lemma 5** *Let an arbitrary on-line algorithm output (not necessarily distinct) hypotheses $H_0, \ldots, H_t$ when it is run on $Z^t$. Then for any $0 < \delta < 1$ the following holds:*

$$\mathbb{P}\left(\min_{0 \le i \le t-1}\left(\mathrm{er}(H_i) + 2\,c_\delta(t-i)\right) \ge \frac{M}{t} + \sqrt{\frac{2}{t}\ln\frac{1}{\delta}} + 4\sqrt{\frac{1}{t}\ln\frac{t+1}{\delta}}\right) \le \delta\,.$$

*Proof.* We have

$$
\begin{aligned}
\min_{0 \le i \le t-1}\left(\mathrm{er}(h_i) + 2\,c_\delta(t-i)\right) &\le \frac{1}{t}\sum_{i=0}^{t-1}\left(\mathrm{er}(h_i) + 2\,c_\delta(t-i)\right)\\
&= \frac{1}{t}\sum_{i=0}^{t-1}\mathrm{er}(h_i) + \frac{2}{t}\sum_{i=0}^{t-1}\sqrt{\frac{1}{2(t-i)}\ln\frac{t(t+1)}{\delta}}\\
&< \frac{1}{t}\sum_{i=0}^{t-1}\mathrm{er}(h_i) + \frac{2}{t}\sum_{i=0}^{t-1}\sqrt{\frac{1}{t-i}\ln\frac{t+1}{\delta}}\\
&\le \frac{1}{t}\sum_{i=0}^{t-1}\mathrm{er}(h_i) + 4\sqrt{\frac{1}{t}\ln\frac{t+1}{\delta}}\,,
\end{aligned}
$$

where the last inequality follows from $\sum_{i=1}^{t}\sqrt{1/i} \le 2\sqrt{t}$. Therefore

$$
\begin{aligned}
\mathbb{P}&\left(\min_{0 \le i \le t-1}\left(\mathrm{er}(H_i) + 2\,c_\delta(t-i)\right) \ge \frac{M}{t} + \sqrt{\frac{2}{t}\ln\frac{1}{\delta}} + 4\sqrt{\frac{1}{t}\ln\frac{t+1}{\delta}}\right)\\
&\le \mathbb{P}\left(\frac{1}{t}\sum_{i=0}^{t-1}\mathrm{er}(H_i) \ge \frac{M}{t} + \sqrt{\frac{2}{t}\ln\frac{1}{\delta}}\right)\\
&\le \delta,
\end{aligned}
$$

by Lemma 1 (with $L = 1$). $\qquad\square$

*Proof (of Theorem 3).* The proof follows by combining Lemma 4 and Lemma 5, and by overapproximating the square root terms therein. $\qquad\square$

## 5  Applications

For the sake of concreteness we now sketch two generalization bounds which can be obtained through a direct application of our techniques.

The $p$-norm Perceptron algorithm [10, 9] is a linear threshold algorithm which keeps in the $i$-th trial a weight vector $\boldsymbol{\theta}_{i-1} \in \mathbb{R}^n$. On instance $\boldsymbol{x}_i \in \mathcal{X} = \{\boldsymbol{x} \in \mathbb{R}^n : \|\boldsymbol{x}\|_p \le 1\}$, the algorithm predicts by $h_{i-1}(\boldsymbol{x}_i) = \mathrm{sign}(\mathbf{g}(\boldsymbol{\theta}_{i-1})\cdot\boldsymbol{x}_i) \in \{-1, +1\}$, where $\mathbf{g}(\boldsymbol{\theta}) = \nabla\frac{1}{2}\|\boldsymbol{\theta}\|_p^2$ and $p \ge 2$. If the algorithm's prediction is wrong (i.e., if $h_{i-1}(\boldsymbol{x}_i) \ne y_i$) then the algorithm performs the weight update $\boldsymbol{\theta}_i \leftarrow \boldsymbol{\theta}_{i-1} + y_i\,\boldsymbol{x}_i$. Notice that $p = 2$ yields the classical Perceptron algorithm [17]. On the other hand, $p = \Theta(\log n)$ gets an algorithm which performs like a multiplicative algorithm, such as the Normalized Winnow algorithm [10]. Applying Theorem 3 to the bound on the number $M$ of mistakes for the $p$-norm Perceptron algorithm shown in [9], we immediately obtain that, with probability at least $1 - \delta$ with respect to the draw of the training sample $Z^t$, the risk $\mathrm{er}(\widehat{H})$ of the penalized estimator $\widehat{H}$ is at most

$$\frac{1}{t}\left(D_\gamma(\boldsymbol{u}, Z^t) + \frac{(p-1)}{\gamma^2} + \frac{1}{\gamma}\sqrt{(p-1)\,D_\gamma(\boldsymbol{u}, Z^t)}\right) + 5\sqrt{\frac{1}{t}\ln\frac{2(t+1)}{\delta}} \qquad (4)$$

for any $\gamma > 0$ and for any $\boldsymbol{u}$ such that $||\boldsymbol{u}||_{p/(p-1)} \leq 1$. The margin-based quantity $D_\gamma(\boldsymbol{u}, z^t) = \sum_{i=1}^{t} \max\{0, 1 - y_i\,\boldsymbol{u}\cdot\boldsymbol{x}_i/\gamma\}$ is called soft margin in [20] and accounts for the distribution of margin values achieved by the examples in $z^t$ with respect to hyperplane $\boldsymbol{u}$. Traditional data-dependent bounds using uniform convergence methods (e.g., [19]) are typically expressed in terms of the sample margin $|\{i\,:\,y_i\,\boldsymbol{u}\cdot\boldsymbol{x}_i \leq \gamma\}|/t$, i.e., in terms of the fraction of training points whose margin is at most $\gamma$. The ratio $D_\gamma(\boldsymbol{u}, z^t)/t$ occurring in (4) has a similar flavor, though the two ratios are, in general, incomparable.

We remark that bound (4) does not have the extra log factors appearing in the analyses based on uniform convergence. Furthermore, it is significantly better than the bound in [20] whenever $D_\gamma/t$ is constant, which typically occurs when the data sequence is not linearly separable.

As a second application, we consider the *ridge regression* algorithm [12] for square loss. Assume $\mathcal{X} = \mathbb{R}^n$ and $\mathcal{Y} = [-Y, +Y]$. This algorithm computes at the beginning of the $i$-th trial the vector $\boldsymbol{w} = \boldsymbol{w}_{i-1}$ which minimizes $\frac{a}{2}||\boldsymbol{w}||_2^2 + \sum_{j=1}^{i-1}\frac{1}{2}(y_j - \boldsymbol{w}\cdot\boldsymbol{x}_j)^2$, where $a > 0$. On instance $\boldsymbol{x}_i$ the algorithm predicts with $h_{i-1}(\boldsymbol{x}_i) = \kappa_Y(\boldsymbol{w}_{i-1}\cdot\boldsymbol{x}_i)$, where $\kappa_Y$ is the "clipping" function $\kappa_Y(\boldsymbol{x}) = Y$ if $\boldsymbol{x} \geq Y$, $\kappa_Y(\boldsymbol{x}) = -Y$ if $\boldsymbol{x} \leq -Y$ and $\kappa_Y(\boldsymbol{x}) = \boldsymbol{x}$ if $-Y \leq \boldsymbol{x} \leq Y$. The losses $\frac{1}{2}(y_i - h_{i-1}(\boldsymbol{x}_i))^2$ are thus bounded by $2\,Y^2$. We can apply Theorem 2 to the bound on the cumulative loss $M$ for ridge regression (see [22, 2]) and obtain that, with probability at least $1 - \delta$ with respect to the draw of the training sample $Z^t$, the risk $\mathrm{er}(\overline{H})$ of the average hypothesis estimator $\overline{H}$ is at most

$$\frac{1}{t}\left(\frac{a}{2}||\boldsymbol{u}||_2^2 + M(\boldsymbol{u}, Z^t) + 2\,Y^2\left(\ln\left|aI + \sum_{i=1}^{t}X_i\,X_i^\top\right| - n\,\ln a\right)\right) + 2\,Y^2\sqrt{\frac{2}{t}\ln\frac{1}{\delta}}$$

(5)

for any $\boldsymbol{u} \in \mathbb{R}^n$, where $M(\boldsymbol{u}, Z^t) = \sum_{i=1}^{t}\frac{1}{2}(Y_i - \boldsymbol{u}\cdot X_i)^2$, $|A|$ denotes the determinant of matrix $A$, $I$ is the $n$-dimensional identity matrix and $A^\top$ is the transpose of $A$.[2] Let us denote by $\boldsymbol{X}_t$ the matrix whose columns are the data vectors $\boldsymbol{x}_i$, $i = 1, \ldots, t$. Then simple linear algebra shows that

$$\ln\left|aI + \sum_{i=1}^{t}X_i\,X_i^\top\right| - n\,\ln a = \ln\left|aI + \boldsymbol{X}_t\boldsymbol{X}_t^\top\right| - n\,\ln a = \sum_{i=1}^{n}\ln\left(1 + \lambda_i/a\right),$$

where the $\lambda_i$'s are the eigenvalues of $\boldsymbol{X}_t\boldsymbol{X}_t^\top$. The nonzero eigenvalues of $\boldsymbol{X}_t\boldsymbol{X}_t^\top$ are the same as the nonzero eigenvalues of the Gram matrix $\boldsymbol{X}_t^\top\boldsymbol{X}_t$. Risk bounds in terms of the eigenvalues of the Gram matrix were also derived in [23]; we defer to the full paper a comparison between these results and ours. Finally, our bound applies also to kernel ridge regression [18] by replacing the eigenvalues of $\boldsymbol{X}_t^\top\boldsymbol{X}_t$ with the eigenvalues of the kernel Gram matrix $K(\boldsymbol{x}_i, \boldsymbol{x}_j)$, $1 \leq i, j \leq t$, where $K$ is the kernel being considered.

## Footnotes

[1]Notice that the last hypothesis $h_t$ is not used in this average.

## References

[1]  Angluin, D. Queries and concept learning, *Machine Learning*, 2(4), 319-342, 1988.

[2]  Azoury, K., and Warmuth, M. K. Relative loss bounds for on-line density estimation with the exponential family of distributions, *Machine Learning*, 43:211–246, 2001.

[3]  K. Azuma. Weighted sum of certain dependend random variables. *Tohoku Mathematical Journal*, 68, 357–367, 1967.

---

[2]Using a slightly different linear regression algorithm, Forster and Warmuth [7] have proven a sharper bound on the *expected* relative loss. In particular, they have exhibited an algorithm computing hypothesis $H = H(Z^t)$ such that in expectation (over $Z^t$) the relative risk $\mathrm{er}(H) - \min_{\boldsymbol{u}\in\mathbb{R}^n}\mathbb{E}[\ell(\boldsymbol{u}\cdot X, Y)]$ is bounded by $n\,Y^2/t$.

[4]  A. Blum, A. Kalai, and J. Langford. Beating the hold-out: bounds for k-fold and progressive cross-validation. In 12th COLT, pages 203–208, 1999.

[5]  S. Boucheron, G. Lugosi, and P. Massart. A sharp concentration inequality with applications. *Random Structures and Algorithms*, 16, 277–292, 2000.

[6]  N. Cesa-Bianchi, Y. Freund, D. Haussler, D. P. Helmbold, R. E. Schapire, and M. K. Warmuth. How to use expert advice. *Journal of the ACM*, 44(3), 427–485, 1997.

[7]  J. Forster, and M. K. Warmuth. Relative expected instantaneous loss bounds. *13th COLT*, 90–99, 2000.

[8]  Y. Freund and R. Schapire. Large margin classification using the perceptron algorithm. *Machine Learning*, 37(3), 277–296, 1999.

[9]  C. Gentile The robustness of the $p$-norm algorithms. Manuscript, 2001. An extended abstract (co-authored with N. Littlestone) appeared in *12th COLT*, 1–11, 1999.

[10]  A. J. Grove, N. Littlestone, and D. Schuurmans. General convergence results for linear discriminant updates, Machine Learning, 43(3), 173–210, 2001.

[11]  D. Helmbold and M. K. Warmuth. On weak learning. *JCSS*, 50(3), 551–573, June 1995.

[12]  A. Hoerl, and R. Kennard, Ridge regression: biased estimation for nonorthogonal problems. *Technometrics*, 12, 55–67, 1970.

[13]  J. Kivinen and M. K. Warmuth. Additive versus exponentiated gradient updates for linear prediction. *Information and Computation*, 132(1), 1–64, 1997.

[14]  N. Littlestone. Learning quickly when irrelevant attributes abound: A new linear-threshold algorithm. *Machine Learning*, 2, 285–318, 1988.

[15]  N. Littlestone. From on-line to batch learning. In *2nd COLT*, 269–284, 1989.

[16]  N. Littlestone and M. K. Warmuth. The weighted majority algorithm. *Information and Computation*, 108(2), 212–261, 1994.

[17]  F. Rosenblatt. *Principles of neurodynamics: Perceptrons and the theory of brain mechanisms.* Spartan Books, Washington, D.C., 1962.

[18]  C. Saunders, A. Gammerman, and V. Vovk. Ridge Regression Learning Algorithm in Dual Variables, In *15th ICML*, 1998.

[19]  J. Shawe-Taylor, P. Bartlett, R. Williamson, and M. Anthony, Structural Risk Minimization over Data-dependent Hierarchies. *IEEE Trans. IT*, 44, 1926–1940, 1998.

[20]  J. Shawe-Taylor and N. Cristianini, On the generalization of soft margin algorithms, 2000. NeuroCOLT2 Tech. Rep. 2000-082, http://www.neurocolt.org.

[21]  V.N. Vapnik, *Statistical learning theory.* J. Wiley and Sons, NY, 1998.

[22]  V. Vovk, Competitive on-line linear regression. In *NIPS\*10*, 1998. Also: Tech. Rep. Department of Computer Science, Royal Holloway, University of London, CSD-TR-97-13, 1997.

[23]  R. C. Williamson, J. Shawe-Taylor, B. Schölkopf and A. J. Smola, Sample Based Generalization Bounds, *IEEE Trans. IT*, to appear.
